# New Algorithms for
# 2D and 3D Point Matching:
# Pose Estimation and Correspondence

**Steven Gold[1], Chien Ping Lu[1], Anand Rangarajan[1],**
**Suguna Pappu[1] and Eric Mjolsness[2]**
Department of Computer Science
Yale University
New Haven, CT 06520-8285

## Abstract

A fundamental open problem in computer vision—determining pose and correspondence between two sets of points in space—is solved with a novel, robust and easily implementable algorithm. The technique works on noisy point sets that may be of unequal sizes and may differ by non-rigid transformations. A 2D variation calculates the pose between point sets related by an affine transformation—translation, rotation, scale and shear. A 3D to 3D variation calculates translation and rotation. An objective describing the problem is derived from Mean field theory. The objective is minimized with clocked (EM-like) dynamics. Experiments with both handwritten and synthetic data provide empirical evidence for the method.

## 1 Introduction

Matching the representations of two images has long been the focus of much research in Computer Vision, forming an essential component of many machine-based ob-

[1] E-mail address of authors: lastname-firstname@cs.yale.edu

[2] Department of Computer Science and Engineering, University of California at San Diego (UCSD), La Jolla, CA 92093-0114. E-mail: emj@cs.ucsd.edu

ject recognition systems. Critical to most matching techniques is the determination of correspondence between spatially localized features within each image. This has traditionally been considered a hard problem - especially when the issues of noise, missing or spurious data, and non-rigid transformations are tackled [Grimson, 1990]. Many approaches have been tried, with tree-pruning techniques and generalized Hough transforms being the most common. We introduce a new, robust and easily implementable algorithm to find such poses and correspondences. The algorithm can determine non-rigid transformations between noisy 2D or 3D spatially located unlabeled feature sets despite missing or spurious features. It is derived by minimizing an objective function describing the problem with a combination of optimization techniques, incorporating Mean Field theory, slack variables, iterative projective scaling, and clocked (EM-like) dynamics.

## 2   2D with Affine Transformations

### 2.1   Formulating the Objective

Our first algorithm calculates the pose between noisy, 2D point sets of unequal size related by an affine transformation - translation, rotation, scale and shear. Given two sets of points $\{X_j\}$ and $\{Y_k\}$, one can minimize the following objective to find the affine transformation and permutation which best maps $Y$ onto $X$ :

$$E_{2D}(m,t,A) = \sum_{j=1}^{J}\sum_{k=1}^{K} m_{jk}\|X_j - t - AY_k\|^2 + g(A) - \alpha\sum_{j=1}^{J}\sum_{k=1}^{K} m_{jk}$$

with constraints: $\forall j \sum_{k=1}^{K} m_{jk} \leq 1$ , $\forall k \sum_{j=1}^{J} m_{jk} \leq 1$ , $\forall jk\ m_{jk} \geq 0$ and

$$g(A) = \gamma a^2 + \kappa b^2 + \lambda c^2$$

$A$ is decomposed into scale, rotation, vertical shear and oblique shear as follows:

$$A = s(a)R(\Theta)Sh_1(b)Sh_2(c)$$

where,

$$s(a) = \begin{pmatrix} e^a & 0 \\ 0 & e^a \end{pmatrix}, \ Sh_1(b) = \begin{pmatrix} e^b & 0 \\ 0 & e^{-b} \end{pmatrix}, \ Sh_2(c) = \begin{pmatrix} \cosh(c) & \sinh(c) \\ \sinh(c) & \cosh(c) \end{pmatrix}$$

$R(\Theta)$ is the standard 2x2 rotation matrix. $g(A)$ serves to regularize the affine transformation - bounding the scale and shear components. $m$ is a fuzzy correspondence matrix which matches points in one image with corresponding points in the other image. The constraints on $m$ ensure that each point in each image corresponds to at most one point in the other image. However, partial matches are allowed, in which case the sum of these partial matches may add up to no more than one. The inequality constraint on $m$ permits a null match or multiple partial matches.

The $\alpha$ term biases the objective towards matches. The decomposition of $A$ in the above is not required, since $A$ could be left as a 2x2 matrix and solved for directly in the algorithm that follows. The decomposition just provides for more precise regularization, i.e., specification of the likely kinds of transformations. Also $Sh_2(c)$ could

be replaced by another rotation matrix, using the singular value decomposition of $A$.

We transform the inequality constraints into equality constraints by introducing slack variables, a standard technique from linear programming;

$$\forall j \sum_{k=1}^{K} m_{jk} \leq 1 \quad \rightarrow \quad \forall j \sum_{k=1}^{K+1} m_{jk} = 1$$

and likewise for the column constraints. An extra row and column are added to the matrix $m$ to hold the slack variables. Following the treatment in [Peterson and Soderberg, 1989; Yuille and Kosowsky, 1994] we employ Lagrange multipliers and an $x \log x$ barrier function to enforce the constraints with the following objective:

$$E_{2D}(m, t, A) = \sum_{j=1}^{J} \sum_{k=1}^{K} m_{jk} \|X_j - t - AY_k\|^2 + g(A) - \alpha \sum_{j=1}^{J} \sum_{k=1}^{K} m_{jk}$$

$$+ \frac{1}{\beta} \sum_{j=1}^{J+1} \sum_{k=1}^{K+1} m_{jk} (\log m_{jk} - 1) + \sum_{j=1}^{J} \mu_j (\sum_{k=1}^{K+1} m_{jk} - 1) + \sum_{k=1}^{K} \nu_k (\sum_{j=1}^{J+1} m_{jk} - 1) \quad (1)$$

In this objective we are looking for a saddle point. (1) is minimized with respect to $m$, $t$, and $A$ which are the correspondence matrix, translation, and affine transform, and is maximized with respect to $\mu$ and $\nu$, the Lagrange multipliers that enforce the row and column constraints for $m$. $m$ is fuzzy, with the degree of fuzziness dependent upon $\beta$.

## 2.2   The Algorithm

The algorithm to minimize the above objective proceeds in two phases. In phase one, while $\{t, A\}$ are held fixed, $m$ is initialized with a coordinate descent step, described below, and then iteratively normalized across its rows and columns until the procedure converges (iterative projective scaling). This phase is analogous to a softmax update, except that instead of enforcing a one-way, winner-take-all (maximum) constraint, a two-way, assignment constraint is being enforced. Therefore we describe this phase as a softassign. In phase two $\{t, A\}$ are updated using coordinate descent. Then $\beta$ is increased and the loop repeats. Let $\hat{E}_{2D}$ be the above objective (1) without the terms that enforce the constraints (i.e. the $x \log x$ barrier function and the Lagrange parameters).

In phase one (softassign) $m$ is updated via coordinate descent:

$$m_{jk} = \exp(-\beta \frac{\partial \hat{E}_{2D}}{\partial m_{jk}})$$

Then $m$ is iteratively normalized across $j$ and $k$ until $\sum_{j=1}^{J} \sum_{k=1}^{K} \Delta m_{iajk} < \epsilon$ :

$$m_{jk} = \frac{m_{jk}}{\sum_{j'=1}^{J+1} m_{j'k}} \quad ; \quad m_{jk} = \frac{m_{jk}}{\sum_{k'=1}^{K+1} m_{jk'}}$$

Using coordinate descent the $\{t, A\}$ are updated in phase two. If a term of $\{A\}$ cannot be computed analytically (because of its regularization), Newton's method

is used to compute the root of the function. So if $a$ is a term of $\{t, A\}$ then in phase two we update $a$ such that $\frac{\partial \hat{E}_{2D}}{\partial a} = 0$. Finally $\beta$ is increased and the loop repeats.

By setting the partial derivatives of $E_{2D}$ to zero and initializing the Lagrange parameters to zero, the algorithm for phase one may be derived. Beginning with a small $\beta$ allows minimization over a fuzzy correspondence matrix $m$, for which a global minimum is easier to find. Raising $\beta$ drives the $m$'s closer to 0 or 1, as the algorithm approaches a saddle point.

## 3   3D with Rotation and Translation

The second algorithm solves the 3D-3D pose estimation problem with unknown correspondence. Given two sets of 3D points $\{X_j\}$ and $\{Y_k\}$ find the rotation $R$, translation $T$, and correspondence $m$ that minimize

$$E_{3D}(m, T, R) = \sum_{j=1}^{J}\sum_{k=1}^{K} m_{jk}\|RX_j + T - Y_k\|^2 - \alpha \sum_{j=1}^{J}\sum_{k=1}^{K} m_{jk}$$

with the same constraint on the fuzzy correspondence matrix $m$ as in 2D affine matching. Note that there is no regularization term for the $T - R$ parameters.

This algorithm also works in two phases. In the first, $m$ is updated by a softassign as was described for 2D affine matching. In the second phase, $m$ is fixed, and the problem becomes a 3D to 3D pose estimation problem formulated as a weighted least squares problem. The rotation and translation are represented by a dual number quaternion $(r, s)$ which corresponds to a screw coordinate transform [Walker et al., 1991]. The rotation can be written as $R(r) = W(r)^t Q(r)$ and the translation as $W(r)^t s$. Using these representations, the objective function becomes

$$E_{3D} = \sum_{j=1}^{J}\sum_{k=1}^{K} m_{jk}\|W(r)^t Q(r)x_j + W(r)^t s - y_k\|^2$$

where $x_j = (X_j, 0)^t$ and $y_k = (Y_k, 0)^t$ are the quaternion representations of $X_j$ and $Y_k$, respectively. Using the properties that $Q(a)b = W(b)a$ and $Q(a)^t Q(a) = W(a)^t W(a) = (a^t a)I$, the objective function can be rewritten as

$$E_{3D} = r^t C_1 r + s^t C_2 s + s^t C_3 r + \lambda_1(r^t r - 1) + \lambda_2(s^t r), \tag{2}$$

where

$$C_1 = -\sum_{j=1}^{J}\sum_{k=1}^{K} m_{jk} Q(y_k)^t W(x_j)$$

$$C_2 = \frac{1}{2}\sum_{j=1}^{J}\sum_{k=1}^{K} m_{jk} I$$

$$C_3 = \sum_{j=1}^{J}\sum_{k=1}^{K} m_{jk}(W(x_j) - Q(y_k)).$$

With this new representation, all the information, including the current fuzzy esti-
mate of the correspondence $m$ are absorbed into the three 4-by-4 matrices $C_1, C_2, C_3$
in (2), which can be minimized in closed-form [Walker et al., 1991].

## 4    Experimental Results

In this section we provide experimental results for both the 2D and 3D matching
problems. As an application of the 2D matching algorithm, we present results in
the context of handwritten character recognition.

### 4.1    Handwritten Character Data

The data were generated using an X-windows tool which enables us to draw an
image with the mouse on a writing pad on the screen. The contours of the images
are discretized and are expressed as a set of points in the plane. In the experiments
below, we generate 70 points per character on average.

The inputs to the point matching algorithm are the x-y coordinates generated by the
drawing program. No other pre-processing is done. The output is a correspondence
matrix and a pose. In Figures 1 and 2, we show the correspondences found between
several images drawn in this fashion.To make the actual point matches easier to
see, we have drawn the correspondences only for every other model point.

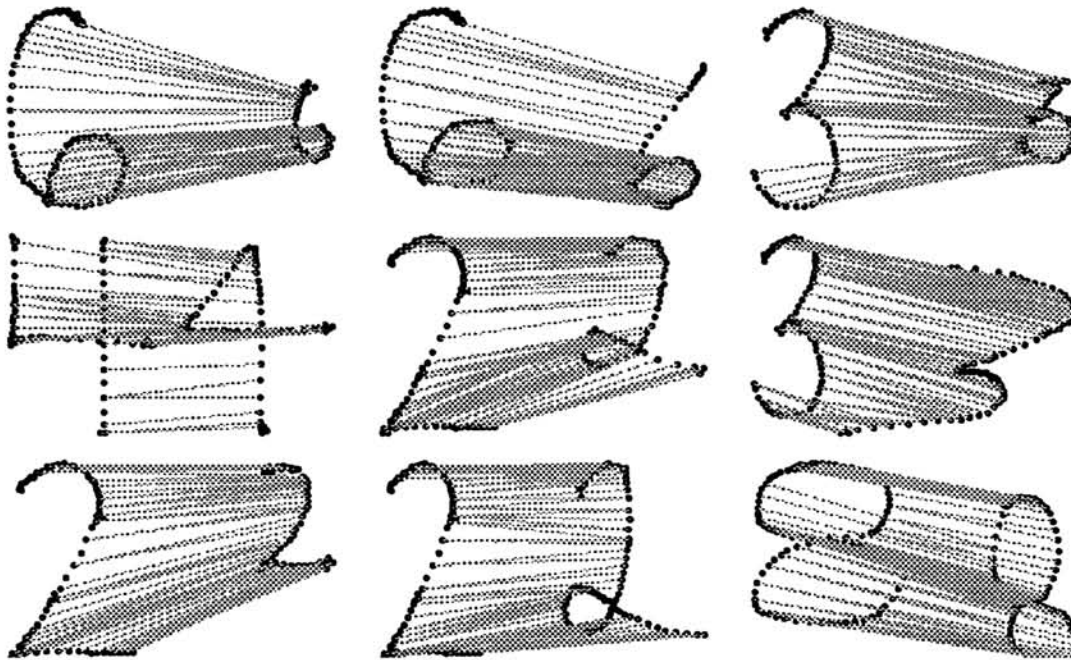

Figure 1: Correspondence of digits

In one experiment, we drew examples of individual digits, one as a model digit
and then many different variations of it. In Figure 1, it can be seen that the

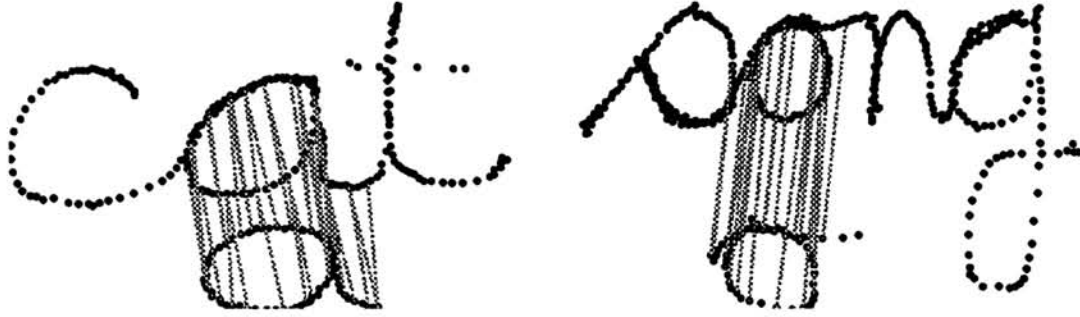

Figure 2: Correspondence: "a" found in "cat", "o" found in "song"

correspondences are good for a large variation from the model digit. For example, the correspondence is invariant to scale. Also, the correspondence is good between distorted digits, as in 3 and 6, or between different forms of a digit as in 4, 3, and 2.

In another experiment (Figure 2), individual letters are correctly identified within words. Here, no pre-processing to segment the cursive word into letters is done. The correspondence returned by the point matching algorithm by itself can be good enough for identification. Even similar letters may be differentiated, for example the "a" in "cat" is correctly identified even though the "c" has a similar shape and the "o" is correctly identified in "song", despite the similarity of the "s".

## 4.2   Randomly generated point sets: 2D

In the second set of experiments, randomly generated dot patterns were used. In each trial a model is created by randomly generating with a uniform distribution, 50 points on a grid of unit area. Independent Gaussian noise $N(0, \sigma)$ is added to each of the points creating a jittered image. Then a fraction, $p_d$, of points are deleted, and a fraction, $p_s$, of spurious points are added, randomly on the unit square. Finally, a randomly generated transformation is applied to the set to generate a new image. The objective then is to recover the transformation and correspondence between the transformed image and the original point set.

The transformations we have considered are $\hat{A} \rightarrow$ (Translation, rotation, scale) and the full affine transformation, $A \rightarrow$ (Translation, rotation, scale, vertical shear, oblique shear) The transformation parameters, $\{t_x, t_y, \theta, a, b, c\}$ are bounded in the following way: $-0.5 < t_x, t_y < 0.5$, $-27° < \theta < 27°$, $0.5 \le e^a \le 2$ where $a$ is the scale parameter, and $0.7 \le e^b, e^c \le 1/0.7$ where $b, c$ are the parameters for the two shears. Each of the parameters is chosen independently and uniformly from the possible ranges.

We use the error measure $e_\alpha = 3|\frac{\alpha^{actual} - \alpha^{estimate}}{width_\alpha}|$ where $e_\alpha$ is the error measure for parameter $\alpha$ and $width_\alpha$ is the range of permissible values for $\alpha$. Dividing by $width_\alpha$ is preferable to dividing by $\alpha^{actual}$, which incorrectly weights small $\alpha^{actual}$ values. The reported error (y axes of Figure 3) is the average error over all the parameters.

The time to recover the correspondence and transformation for a problem instance of 50 points is about 50 seconds on a Silicon Graphics workstation with a R4400 processor. By varying parameters such as the annealing rate or stopping criterion, this can be reduced to about 20 seconds with some degradation in accuracy. For each trial combinations of $\sigma \in \{0.01, 0.02, \ldots, 0.08\}$ and $p_d \in \{0\%, 10\%, 30\%, 50\%\}$ and $p_s \in \{0\%, 10\%\}$ were used.

Results are reported separately for transformations $\hat{A}$ and $A$. For each combination of $(\sigma, p_d, p_s)$ 500 test instances were generated. Each data point in Figures 3.a and 3.b represents the average error measure for these 500 experiments. The noise and/or deletion-addition factor increases the error measure monotonically. As expected, the transformation $\hat{A}$ has better results than the affine transformation $A$.

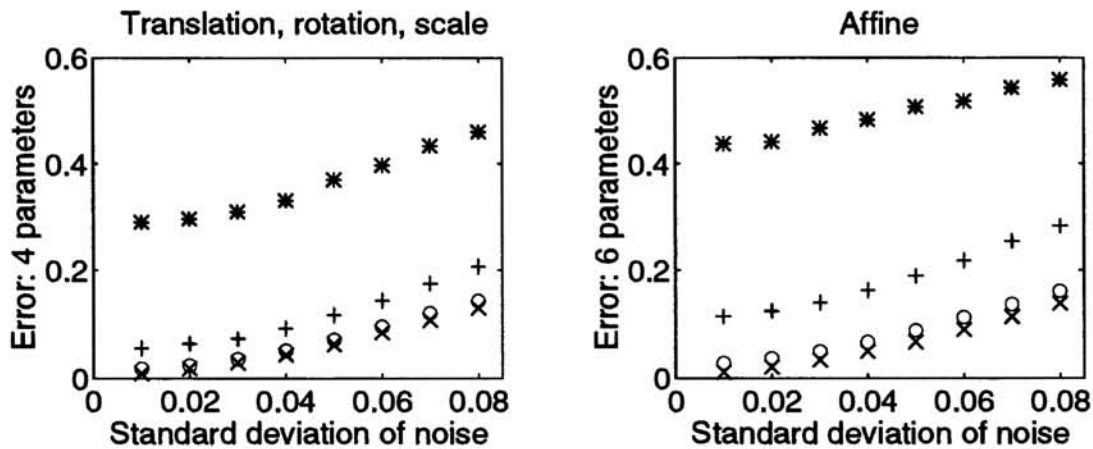

Figure 3: 2D Results for Synthetic Data

$x : p_d = 0.0, p_s = 0.0,$     $o : p_d = 0.1, p_s = 0.1$
$+ : p_d = 0.3, p_s = 0.1,$     $* : p_d = 0.5, p_s = 0.1$

## 4.3   Randomly generated point sets: 3D

A test instance for 3D point matching involves generating a random 3D point set as a model image, and then generating a test image by applying a random transformation, adding noise and then randomly deleting points.

20 points are generated uniformly within an unit cube. The parameters for the transformation are generated as follows: The three rotation angles for $R$ are selected from a uniform distribution $U[20, 70]$. Translation parameters $T_x, T_y, T_z$ are selected from a uniform distribution $U[2.5, 7.5]$. Gaussian noise $N(0, \sigma)$ is added to the points. The objective then is to recover the three translation and three rotation parameters and to find the correspondence between this and the original point set. The results are summarized in Figure 4.

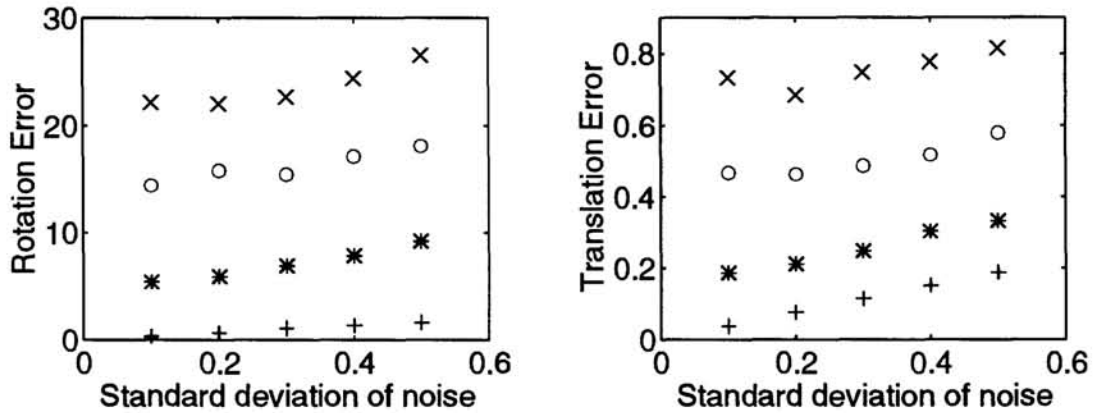

Figure 4: 3D Results for Synthetic Data

$x : p_d = 0.0, p_s = 0.0,$      $o : p_d = 0.1, p_s = 0.1$
$+ : p_d = 0.2, p_s = 0.2,$      $* : p_d = 0.3, p_s = 0.3$

## 5    Conclusion

We have developed an algorithm for solving 2D and 3D correspondence problems. The algorithm handles significant noise, missing or spurious features, and non-rigid transformations. Moreover it works with point feature data alone; inclusion of other types of feature information could improve its accuracy and speed. This approach may also be extended to solve multi-level problems. Additionally, the affine transform might be modified to include higher order transformations. It may also be used as a distance measure in learning [Gold et al.,1994].

### Acknowledgements

This work has been supported by AFOSR grant F49620-92-J-0465, ONR/DARPA grant N00014-92-J-4048, and the Yale Center for Theoretical and Applied Neuroscience (CTAN). Jing Yan developed the handwriting interface.

### References

S. Gold, E. Mjolsness and A. Rangarajan. (1994) Clustering with a domain-specific distance measure. In J.D. Cowan et al., (eds.), *NIPS 6*. Morgan Kaufmann.

E. Grimson, (1990) *Object Recognition by Computer*, Cambridge, MA: MIT Press

C. Peterson and B. Söderberg. (1989) A new method for mapping optimization problems onto neural networks, *Int. Journ. of Neural Sys.*, **1**(1):3:22.

M. W. Walker, L. Shao and R. Volz. (1991) *Estimating 3-D location parameters using dual number quaternions*, CVGIP: Image Understanding *54*(3):358-367.

A. L. Yuille and J. J. Kosowsky. (1994). Statistical physics algorithms that converge. *Neural Computation*, **6**:341-356.